# Summed Weight Neuron Perturbation: An O(N) Improvement over Weight Perturbation.

**Barry Flower and Marwan Jabri**

SEDAL
Department of Electrical Engineering
University of Sydney
NSW 2006 Australia

## Abstract

The algorithm presented performs gradient descent on the weight space of an Artificial Neural Network (ANN), using a finite difference to approximate the gradient. The method is novel in that it achieves a computational complexity similar to that of Node Perturbation, $O(N^3)$, but does not require access to the activity of hidden or internal neurons. This is possible due to a stochastic relation between perturbations at the weights and the neurons of an ANN. The algorithm is also similar to Weight Perturbation in that it is optimal in terms of hardware requirements when used for the training of VLSI implementations of ANN's.

## 1  INTRODUCTION

Optimization of the weights of an ANN may be performed by, the application of a gradient descent technique. The gradient may be calculated directly as in Backpropagation, or it may be approximated by a Finite Difference Method which is what we concern ourselves with in this paper. These methods lend themselves to the task of training hardware implementations of ANNs where real estate is at a premium and synaptic density is of great importance. Neuron Perturbation (NP), as described by the Madaline Rule III (MRIII) (Widrow and Lehr, 1990), is a technique that approximates the gradient of the Mean Square Error (MSE) *with respect to* the change at a given neuron by applying a small perturbation to the input of the neuron and measuring the change in the MSE. The weight

$$\Delta w_{ij} = -\eta \cdot \frac{\partial E}{\partial net_i} \cdot x_j, \tag{1}$$

update is then calculated from the product of this gradient measure and the activation of

the neuron from which the weight is fed, as described by (1).

Weight Perturbation (WP), as described by Jabri and Flower (Jabri and Flower, 1992) is a neural network training techniques based on gradient descent using a Finite Difference method to approximate the gradient. The gradient of the MSE *with respect to* a weight is approximated by applying a small pertubation to the weight and measuring the change in the MSE. This gradient is then used to calculated the weight update such that:

$$\Delta w_{ij} = -\eta \cdot \frac{\partial E}{\partial w_{ij}} \tag{2}$$

The advantages of WP over NP are that it performs better when limited precision weights are used, as shown by Xie and Jabri (Xie and Jabri, 1992), and is optimal with respect to hardware requirements when used to train VLSI implementations of ANNs. However, WP has $O(N^4)$ computational complexity whilst NP has $O(N^3)$ computational complexity.

Summed Weight Neuron Perturbation (SWNP) is similar to NP in that it has a computational complexity of $O(N^3)$ but it has the added advantage that the activation of internal neurons does not need to be known. The cost of this reduced computational complexity is that SWNP needs to save the perturbation vector used.

In the following sections a description of the SWNP algorithm is provided and, finally, some experimental results are presented.

## 2   THE SUMMED WEIGHT NEURON PERTURBATION ALGORITHM

A subsection of a feedforward ANN containing $N$ neurons is shown in Figure 1. on which nomenclature the following derivation is based.

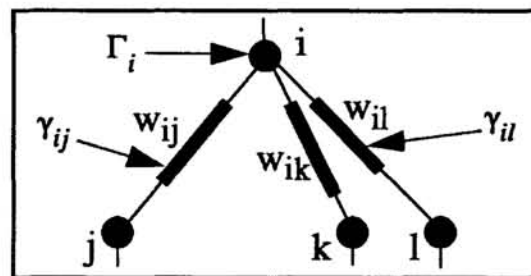

**FIGURE 1:**   Description Of Indices Used To Describe The Neurons Weights And Perturbations In An ANN.

In a feedforward network of size $N$ neurons the activation of a given neuron is determined by:

$$x_i(p) = f_i(net_i(p)), \quad \text{and} \quad net_i(p) = \sum_l w_{il} x_l(p), \tag{3}$$

and $f_i(y)$ is the $i$th neuron transfer function, $x_i(p)$ is the activation of the $i$th neuron for the $p$th pattern, and $w_{il}$ is the weight connecting the $l$th neuron's output to the $i$th neuron's input. The error function, (MSE), is defined as in (4), where $T$ is the set of output neurons and $d_k(p)$ is the expected value of the output on the $k$th neuron. The change in $E(p)$

*with respect to* a given weight may then be expressed as (5).

$$E(p) = \frac{1}{2} \sum_{k \in T} (d_k(p) - x_k(p))^2. \tag{4}$$

$$\frac{\partial E(p)}{\partial w_{ij}} = \frac{\partial E(p)}{\partial net_i(p)} \cdot x_j(p). \tag{5}$$

The first term of on the right-hand side of (5) can be determined using a Finite Difference, which in this case is a Forward Difference, so that:

$$\frac{\partial E(p)}{\partial net_i(p)} = \frac{\Delta E_{\Gamma_i}(p)}{\Gamma_i} + O(\Gamma_i), \tag{6}$$

where,

$$\Delta E_{\Gamma_i}(p) = E_{\Gamma_i}(p) - E(p), \tag{7}$$

and $\Gamma_i$ is the perturbation applied to the $i$th neuron, $E_{\Gamma_i}(p)$ is the error for the $p$th pattern with a perturbation applied to the $i$th neuron and $E(p)$ is the error for the $p$th pattern without a perturbation applied to any neurons. The error introduced by the approximation is represented by the last term on the right-hand side in (6).

The perturbation of one or more of the weights that are inputs to the $q$th neuron can be thought of as being equal to some perturbation applied directly to that neuron. Hence:

$$\Gamma_q = \sum_T \gamma_{ql} x_l(p), \tag{8}$$

where $\gamma_{ql}$ is the perturbation applied to weight $w_{ql}$. As will be shown, perturbing the $q$th neuron by perturbing all the weights feeding into it, enables the sign of the gradient $\frac{\partial E(p)}{\partial w_{ij}}$ to be determined without performing the product on the right-hand side of (5). Further more, the activation of hidden neurons, (i.e. $x_j(p)$ in (5)) need not be known. The contribution of the perturbation of weight $w_{ij}$ to the perturbation of the $i$th neuron is

$$\gamma_{ij} x_j(p). \tag{9}$$

Let us take the degenerate case where there is only one weight for the $i$th neuron. Then the gradient of the MSE *with respect to* weight $w_{ij}$ is:

$$\frac{\partial E(p)}{\partial w_{ij}} = \frac{\Delta E_{\Gamma_i}(p) x_j(p)}{\Gamma_i} + O(\Gamma_i) = \frac{\Delta E_{\Gamma_i}(p) x_j(p)}{\gamma_{ij} x_j(p)} + O(\Gamma_i)$$

$$= \frac{\Delta E_{\Gamma_i}(p)}{\gamma_{ij}} + O(\Gamma_i), \tag{10}$$

noting that $x_j(p)$ has been eliminated. In the general case where the $i$th neuron has more than one weight the gradient *with respect to* weight $w_{ij}$ is shown in (11).

$$\frac{\partial E(p)}{\partial w_{ij}} = \frac{\Delta E_{\Gamma_i}(p)\, x_j(p)}{\Gamma_i} + O(\Gamma_i)$$

$$= \frac{\Delta E_{\Gamma_i}(p)}{\Psi_{ij}} + O(\Gamma_i) \tag{11}$$

where,

$$\Psi_{ij} = \frac{\Gamma_i}{x_j(p)}. \tag{12}$$

The form of (10) and (11) are the same and it will be shown that $\gamma_{ij}$ can be substituted for $\Psi_{ij}$ in (11) due to a stochastic relationship between them.

Let us represent the sign of $\gamma_{ij}$ and $\Psi_{ij}$ as either +1 or -1 such that:

$$\mu_{ij} = \frac{|\gamma_{ij}|}{\gamma_{ij}} \quad \text{and} \quad \nu_{ij} = \frac{|\Psi_{ij}|}{\Psi_{ij}}. \tag{13}$$

The set of all possible states for the system represented by the vector $(\mu_{ij}, \nu_{ij})$, assuming $\gamma_{ij}$ and $\Psi_{ij}$ are never zero, is:

$$\{\,(-1,-1),\ (-1,1),\ (1,-1),\ (1,1)\,\}. \tag{14}$$

and it can be seen that when $\mu_{ij} = \nu_{ij}$ then the sign of the gradient of the MSE *with respect to* weight $w_{ij}$ given by (10) is the same as that given by (11). If the sign of $\gamma_{ij}$ is chosen randomly then the probability of $\mu_{ij} = \nu_{ij}$ being true is 0.5, from (14), and so (10) will generate a gradient that is in the correct direction 50% of the time. This in itself is not sufficient to allow the network to be trained as it will take as many steps in the incorrect direction as the correct direction if the steps themselves are of the same size, (i.e. the magnitude of $\Gamma_i$ is the same for a step in the correct direction as a step in the incorrect direction).

Fortunately it can be shown that the size of the steps in the correct direction are greater than those in the incorrect direction. Let us take the case where a particular $\gamma_{ij}$ is chosen such that

$$\mu_{ij} = \nu_{ij}. \tag{15}$$

Now by substituting (8), (12) and (13) into (15) we get:

$$\frac{|\gamma_{ij}|}{\gamma_{ij}} = \frac{\left|\frac{\sum_k \gamma_{ik}x_k(p)}{x_j}\right|}{\frac{\sum_k \gamma_{ik}x_k(p)}{x_j}} \tag{16}$$

rearranging to give,

$$\frac{|\gamma_{ij}x_j|}{\gamma_{ij}x_j} = \frac{\left|\sum_k \gamma_{ik}x_k(p)\right|}{\sum_k \gamma_{ik}x_k(p)}, \tag{17}$$

which implies that the contribution to $\Gamma_i$ made by the perturbation $\gamma_{ij}$ is of the same sign as $\Gamma_i$. Let us designate this neuron perturbation as $\Gamma_i(A)$. Now we take the other possible case where,

$$\mu_{ij} \neq \nu_{ij}, \tag{18}$$

assuming every other parameter is the same, and only the sign of $\gamma_{ij}$ is changed. The equality in (17) is now untrue and the contribution to $\Gamma_i$ made by the perturbation $\gamma_{ij}$ is of the opposite sign as $\Gamma_i$. Let us designate this neuron perturbation as $\Gamma_i(B)$. From (8) we can determine that,

$$|\Gamma_i(A)| = |\Gamma_i(B)| + 2|\gamma_{ij}x_j|. \tag{19}$$

Equation (19) shows the relationship between the two possible states of the system where $\Gamma_i(A)$ represents the summed neuron perturbation for a selected weight perturbation $\gamma_{ij}$ that generates a step in the corrected direction and $\Gamma_i(B)$ is similar but for a step in the incorrect direction. Clearly the correct step is always calculated from an approximated gradient that is larger than that for an incorrect step as the neuron perturbation is larger. The weight update rule then becomes:

$$\Delta w_{ij} = -\eta \cdot \frac{\Delta E_{\Gamma_i}(p)}{\gamma_{ij}}. \tag{20}$$

The algorithm for SWNP is shown as pseudo code in Figure 2.

## 2.1  HARDWARE COMPATIBILITY OF SWNP

This optimisation technique is ideally suited to the training of hardware implementations of ANN's whether they consist of discrete components or are VLSI technology. The speed up over WP of $O(N)$ achieved is at the cost of an $O(N)$ storage requirement but this storage can be achieved with a single bit per neuron. SWNP is the same order of complexity as NP but does not require access to the activation of internal neurons and therefore can treat a network as a "black box" into which an input vector and weight matrix is fed and an

output vector is received.

```
While (total error > error threshold) {
    For (all patterns in training set) {
        Select next pattern and training vector;
        Forward Prop.;Measure, (calculate) and save error;
        Accumulate total error;
        For (all non-input neurons) {
            For (all weights of current neuron) {
                Apply & Save perturbation of random polarity;
            }
            Forward Prop.;Measure, (calculate) and save Δerror;
            For (all weights of current neuron) {
                Restore value of weight;
                Calculate weight delta using saved perturbation value;
                If (Online Mode) Update current weight;
            }
            If (Online Mode)
                Forward Prop.; Measure, (calculate) and save new error;
        }
        If (Batch Mode) {
            For (all weights)
                Update current weight;
        }
    }
}
```

**FIGURE 2:**   Algorithm in Pseudo Code for Summed Weight Neuron Perturbation.

## 3   TEST RESULTS USING SWNP

The results for a series of tests are shown in the next three tables and are summarised in Figure 4. The headings are, **N** the number of neurons in the network, **P** the number of patterns in the training set, **FF-SWNP** the number of feedforward passes for the SWNP Algorithm, **FF-WP** the number of feedforward passes for the WP Algorithm, and **RATIO** the ratio between the number of feedforward passes for WP against SWNP. The feedforward passes are recorded to 1 significant figure.

The results for a series of simulations comparing the performance of SWNP against WP are shown in Table 1. The simulations utilised floating point synaptic and neuron precisions.

The results for a series of simulations comparing the performance of SWNP against WP are shown in Table 2. The simulations utilised limited synaptic precision, (i.e. 6 bits) and floating point neuron precisions

The results for a series of experiments comparing the performance of SWNP against WP are shown in Table 2. Note: the training algorithm are the variations of WP and SWNP that are combined with the Random Search Algorithm (RSA). The results reported are averaged over 10 trials.

An example of the training error trajectories of WP and SWNP for the Monk 2 problem are shown in Figure 3.

**Table 1: Performance Of SWMP Versus WP, Comparing Feedforward Operations To Convergence. (Simulations With Floating Point Precision)**

| PROBLEM | N | P | FF-SWNP | FF-WP | ERROR | RATIO |
|---|---|---|---|---|---|---|
| XOR | 3 | 4 | $1.6 \times 10^3$ | $1.9 \times 10^3$ | 0.0125 | 1. 22 |
| 4 Encoder | 5 | 4 | $0.9 \times 10^3$ | $1.8 \times 10^3$ | 0.0125 | 1.84 |
| 8 Encoder | 11 | 8 | $1.5 \times 10^5$ | $4.5 \times 10^5$ | 0.0125 | 2.88 |
| ICEG | 15 | 119 | $3.7 \times 10^5$ | $7.9 \times 10^6$ | 0.0125 | 21.34 |

**Table 2: Performance Of SWNP Versus WP, Comparing Feedforward Operations To Convergence. (Simulations With Limited Precision)**

| PROBLEM | N | P | SWNP | WP | ERROR | RATIO |
|---|---|---|---|---|---|---|
| Monk1 | 4 | 129 | $1.0 \times 10^5$ | $1.9 \times 10^6$ | 0.001 | 19.38 |
| Monk2 | 17 | 169 | $3.6 \times 10^5$ | $6.8 \times 10^6$ | 0.0005 | 18.71 |
| Monk3 | 17 | 122 | $1.2 \times 10^6$ | $7.1 \times 10^6$ | 0.022 | 5.87 |
| IECG 55 | 5 | 8 | $1.6 \times 10^4$ | $7.2 \times 10^4$ | 0.0001 | 4.2 |

**Table 3: Performance Of SWNP Versus WP, Comparing Feedforward Operations To Convergence. (Hardware Implementation)**

| PROBLEM | N | P | SWNP | WP | ERROR | RATIO |
|---|---|---|---|---|---|---|
| ECG 55 | 5 | 8 | $3.1 \times 10^3$ | $3.6 \times 10^3$ | 0.00001 | 1.13 |
| ECG 045 | 5 | 8 | $1.1 \times 10^4$ | $2.0 \times 10^4$ | 0.001 | 1.78 |

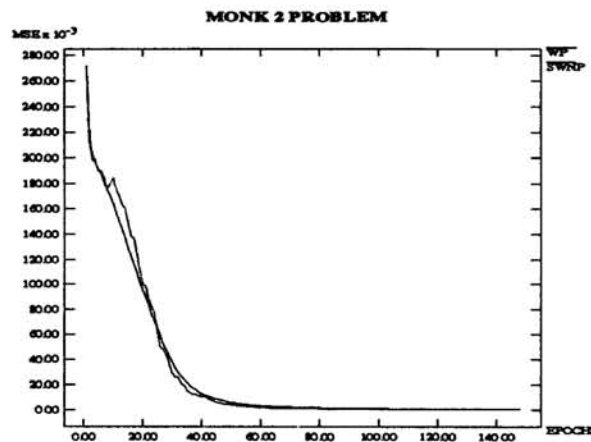

**FIGURE 3:** Comparison of WP and SWNP For Monk 2 Problem

**FIGURE 4:** Comparison of the number of Feedforward passes performed to achieve convergence on a range of problems using SWNP and WP.

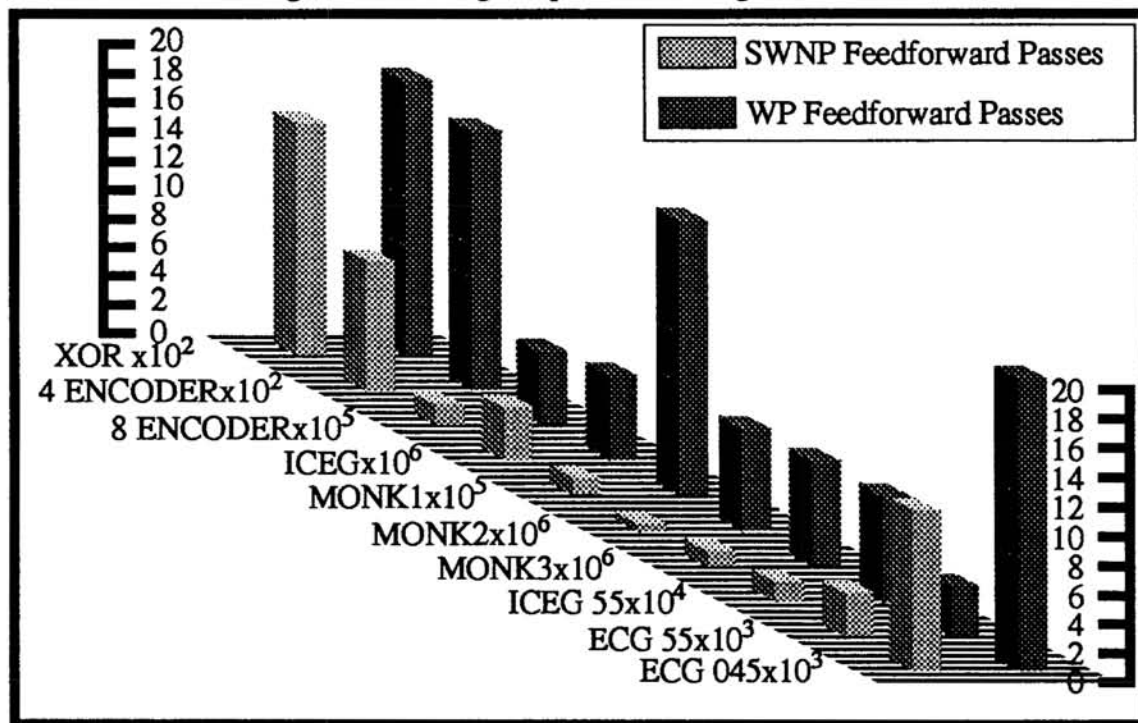

## 4  CONCLUSION

The algorithm presented, SWNP, performs gradient descent on the weight space of an ANN, using a finite difference to approximate the gradient. The method is novel in that it achieves $O(N^3)$ computational complexity similar to that of Node Perturbation but does not require access to the activity of hidden or internal neurons. The algorithm is also similar to Weight Perturbation in that it is optimal in terms of hardware requirements when used for the training of VLSI implementations of ANN's. Results are presented that show the algorithm in operation on floating point simulations, limited precision simulations and an actual hardware implementation of an ANN.

### References

Jabri, M. and Flower, B. (1992). Weight perturbation: An optimal architecture and learning technique for analog vlsi feedforward and recurrent multilayer networks. *IEEE Transactions on Neural Networks*, 3(1):154–157.

Widrow, B. and Lehr, M. A. (1990). 30 years of adaptive neural networks: Perceptron, madaline, and backpropagation. *Proceedings of the IEEE*, 78(9):1415–1442.

Xie, Y. and Jabri, M. (1992). Analysis of the effects of quantization in multilayer neural networks using a statistical model. *IEEE Transactions on Neural Networks*, 3(2):334–338.
